# Shifting, One-Inclusion Mistake Bounds and Tight Multiclass Expected Risk Bounds

**Benjamin I. P. Rubinstein**
Computer Science Division
University of California, Berkeley
Berkeley, CA 94720-1776, U.S.A.
benr@cs.berkeley.edu

**Peter L. Bartlett**
Computer Science Division and
Department of Statistics
University of California, Berkeley
bartlett@cs.berkeley.edu

**J. Hyam Rubinstein**
Department of Mathematics & Statistics
The University of Melbourne
Parkville, Victoria 3010, Australia
rubin@ms.unimelb.edu

## Abstract

Under the prediction model of learning, a prediction strategy is presented with an i.i.d. sample of $n - 1$ points in $\mathcal{X}$ and corresponding labels from a concept $f \in \mathcal{F}$, and aims to minimize the worst-case probability of erring on an $n^{\text{th}}$ point. By exploiting the structure of $\mathcal{F}$, Haussler *et al.* achieved a $\mathrm{VC}(\mathcal{F})/n$ bound for the natural one-inclusion prediction strategy, improving on bounds implied by PAC-type results by a $O(\log n)$ factor. The key data structure in their result is the natural subgraph of the hypercube—the one-inclusion graph; the key step is a $d = \mathrm{VC}(\mathcal{F})$ bound on one-inclusion graph density. The first main result of this paper is a density bound of $n \left( \begin{smallmatrix} n-1 \\ \leq d-1 \end{smallmatrix} \right) / \left( \begin{smallmatrix} n \\ \leq d \end{smallmatrix} \right) < d$, which positively resolves a conjecture of Kuzmin & Warmuth relating to their unlabeled Peeling compression scheme and also leads to an improved mistake bound for the randomized (deterministic) one-inclusion strategy for all $d$ (for $d \approx \Theta(n)$). The proof uses a new form of VC-invariant shifting and a group-theoretic symmetrization. Our second main result is a $k$-class analogue of the $d/n$ mistake bound, replacing the VC-dimension by the Pollard pseudo-dimension and the one-inclusion strategy by its natural hypergraph generalization. This bound on expected risk improves on known PAC-based results by a factor of $O(\log n)$ and is shown to be optimal up to a $O(\log k)$ factor. The combinatorial technique of shifting takes a central role in understanding the one-inclusion (hyper)graph and is a running theme throughout.

## 1 Introduction

In [4, 3] Haussler, Littlestone and Warmuth proposed the one-inclusion prediction strategy as a natural approach to the prediction (or mistake-driven) model of learning, in which a prediction strategy maps a training sample and test point to a test prediction with hopefully guaranteed low probability of erring. The significance of their contribution was two-fold. On the one hand the derived $\mathrm{VC}(\mathcal{F})/n$ upper-bound on the worst-case expected risk of the one-inclusion strategy learning from $\mathcal{F} \subseteq \{0, 1\}^{\mathcal{X}}$ improved on the PAC-based previous-best by an order of $\log n$. This was achieved by taking the structure of the underlying $\mathcal{F}$ into account—which had not been done in previous work—in order to break ties between hypotheses consistent with the training set but offering contradictory predictions on a given test point. At the same time Haussler [3] introduced the idea of *shifting* sub-

sets of the $n$-cube down around the origin—an idea previously developed in Combinatorics—as a powerful tool for learning-theoretic results. In particular, shifting admitted deeply insightful proofs of Sauer's Lemma and a VC-dimension bound on the density of the one-inclusion graph—the key result needed for the one-inclusion strategy's expected risk bound. Recently shifting has impacted on work towards the sample compressibility conjecture of [7] e.g. in [5].

Here we continue to study the one-inclusion graph—the natural graph structure induced by a subset of the $n$-cube—and its related prediction strategy under the lens of shifting. After the necessary background, we develop the technique of shatter-invariant shifting in Section 3. While a subset's VC-dimension cannot be increased by shifting, shatter-invariant shifting guarantees a finite sequence of shifts to a fixed-point under which the shattering of a chosen set remains invariant, thus preserving VC-dimension throughout. In Section 4 we apply a group-theoretic symmetrization to tighten the mistake bound—the worst-case expected risk bound—of the deterministic (randomized) one-inclusion strategy from $d/n$ to $\lceil D_n^d \rceil/n$ ($D_n^d/n$), where $D_n^d < d$ for all $n, d$. The derived $D_n^d$ density bound positively resolves a conjecture of Kuzmin & Warmuth which was suggested as a step towards a correctness proof of the Peeling compression scheme [5]. Finally we generalize the prediction model, the one-inclusion strategy and its bounds from binary to $k$-class learning in Section 5. Where $\Psi_{\mathrm{G}}\text{-dim}\,(\mathcal{F})$ and $\Psi_{\mathrm{P}}\text{-dim}\,(\mathcal{F})$ denote the Graph and Pollard dimensions of $\mathcal{F}$, the best bound on expected risk for $k \in \mathbb{N}$ to-date is $O(\alpha \log \alpha)$ for $\alpha = \Psi_{\mathrm{G}}\text{-dim}\,(\mathcal{F})/n$, for consistent learners [8, 1, 2, 4]. For large $n$ this is $O(\log n \Psi_{\mathrm{G}}\text{-dim}\,(\mathcal{F})/n)$; we derive an improved bound of $\Psi_{\mathrm{P}}\text{-dim}\,(\mathcal{F})/n$ which we show is at most a $O(\log k)$ factor from optimal. Thus, as in the binary case, exploiting class structure enables significantly better bounds on expected risk for multiclass prediction.

As always some proofs have been omitted in the interest of flow or space. In such cases see [8].

## 2   Definitions & background

In this paper sets/random variables, scalars and vectors will be written in uppercase, lowercase and bolded typeface as in $C, x, \mathbf{v}$. We define $\binom{n}{\leq r} = \sum_{i=0}^{r} \binom{n}{i}$, $[n] = \{1, \ldots, n\}$ and $S_n$ to be the set of permutations on $[n]$. We write the density of graph $G = (V, E)$ as $\text{dens}\,(G) = |E|/|V|$, the indicator of $A$ as $\mathbf{1}\,[A]$, and $\exists! x \in X, P(x)$ to mean "there exists a unique $x \in X$ satisfying $P$."

### 2.1   The prediction model of learning

We begin with the basic setup of [4]. Set $\mathcal{X}$ is the *domain* and $\mathcal{F} \subseteq \{0, 1\}^{\mathcal{X}}$ is a *concept class* on $\mathcal{X}$. For notational convenience we write $\text{sam}\,(\mathbf{x}, f) = ((x_1, f(x_1)), \ldots, (x_n, f(x_n)))$ for $\mathbf{x} \in \mathcal{X}^n$, $f \in \mathcal{F}$. A *prediction strategy* is a mapping of the form $Q : \bigcup_{n>1} (\mathcal{X} \times \{0, 1\})^{n-1} \times \mathcal{X} \to \{0, 1\}$.

**Definition 2.1** *The* prediction model *of learning concerns the following scenario. Given full knowledge of strategy $Q$, an adversary picks a distribution $P$ on $\mathcal{X}$ and concept $f \in \mathcal{F}$ so as to maximize the probability of $\{Q\,(\text{sam}\,(X_1, \ldots, X_{n-1}, f), X_n) \neq f(X_n)\}$ where $X_i \stackrel{i.i.d.}{\sim} P$. Thus the measure of performance is the worst-case expected risk*

$$\hat{M}_{Q,\mathcal{F}}(n) = \sup_{f \in \mathcal{F}} \sup_{P} \mathbb{E}_{\mathbf{X} \sim P^n} \left[ \mathbf{1}\,[Q\,(\text{sam}\,((X_1, \ldots, X_{n-1}), f), X_n) \neq f(X_n)] \right] .$$

*A* mistake bound *for $Q$ with respect to $\mathcal{F}$ is an upper-bound on $\hat{M}_{Q,\mathcal{F}}$.*

In contrast to Valiant's PAC model, the prediction learning model is not interested in approximating $f$ given an $f$-labeled sample, but instead in predicting $f(X_n)$ with small worst-case probability of erring. The following allows us to derive mistake-bounds by bounding a worst-case average.

**Lemma 2.2 (Corollary 2.1 [4])** *For any $n > 1$, concept class $\mathcal{F}$ and prediction strategy $\mathcal{Q}$,*

$$\hat{M}_{Q,\mathcal{F}}(n) \leq \sup_{f \in \mathcal{F}} \sup_{\mathbf{x} \in \mathcal{X}^n} \frac{1}{n!} \sum_{g \in S_n} \mathbf{1}\,\left[ Q\,\left(\text{sam}\,\left((x_{g(1)}, \ldots, x_{g(n-1)}), f\right), x_{g(n)}\right) \neq f\left(x_{g(n)}\right) \right]$$

$$= \hat{\bar{M}}_{Q,\mathcal{F}}(n) .$$

*A* permutation mistake bound *for $Q$ with respect to $\mathcal{F}$ is an upper-bound on $\hat{\bar{M}}_{Q,\mathcal{F}}$.*

## 2.2 The capacity of function classes contained in $\{0,\ldots,k\}^{\mathcal{X}}$

We denote by $\Pi_{\mathbf{x}}\left(\mathcal{F}\right) = \{(f(x_1),\ldots,f(x_n)) \mid f \in \mathcal{F}\}$ the projection of $\mathcal{F} \subseteq \mathcal{Y}^{\mathcal{X}}$ on $\mathbf{x} \in \mathcal{X}^n$.

**Definition 2.3** *The* Vapnik-Chervonenkis dimension *of concept class $\mathcal{F}$ is defined as* $\mathrm{VC}(\mathcal{F}) = \sup\{n \mid \exists \mathbf{x} \in \mathcal{X}^n, \Pi_{\mathbf{x}}\left(\mathcal{F}\right) = \{0,1\}^n\}$. *An $\mathbf{x}$ witnessing $\mathrm{VC}(\mathcal{F})$ is said to be* shattered *by $\mathcal{F}$.*

**Lemma 2.4 (Sauer's Lemma [9])** *For any $n \in \mathbb{N}$ and $V \subseteq \{0,1\}^n$, $|V| \leq \binom{n}{\leq \mathrm{VC}(V)}$. A subset $V$ meeting this with equality is called* maximum*.*

It is well-known that the VC-dimension is an inappropriate measure of capacity when $|\mathcal{Y}| > 2$. The following unifying framework of class capacities for $|\mathcal{Y}| < \infty$ is due to [1].

**Definition 2.5** *Let $k \in \mathbb{N}$, $\mathcal{F} \subseteq \{0,\ldots,k\}^{\mathcal{X}}$ and $\Psi$ be a family of mappings $\psi : \{0,\ldots,k\} \to \{0,1,*\}$ called* translations. *For $\mathbf{x} \in \mathcal{X}^n$, $\mathbf{v} \in \Pi_{\mathbf{x}}\left(\mathcal{F}\right) \subseteq \{0,\ldots,k\}^n$ and $\psi \in \Psi^n$ we write $\psi(\mathbf{v}) = (\psi_1(v_1),\ldots,\psi_n(v_n))$ and $\psi(\Pi_{\mathbf{x}}\left(\mathcal{F}\right)) = \{\psi(\mathbf{v}) : \mathbf{v} \in \Pi_{\mathbf{x}}\left(\mathcal{F}\right)\}$. $\mathbf{x} \in \mathcal{X}^n$ is $\Psi$-shattered by $\mathcal{F}$ if there exists a $\psi \in \Psi^n$ such that $\{0,1\}^n \subseteq \psi(\Pi_{\mathbf{x}}\left(\mathcal{F}\right))$. The $\Psi$-dimension of $\mathcal{F}$ is defined by $\Psi\text{-dim}\left(\mathcal{F}\right) = \sup\{n \mid \exists \mathbf{x} \in \mathcal{X}^n, \psi \in \Psi^n \ s.t. \ \{0,1\}^n \subseteq \psi(\Pi_{\mathbf{x}}\left(\mathcal{F}\right))\}$.*

We next describe three important translation families used in this paper.

**Example 2.6** *The families $\Psi_P = \{\psi_{P,i} : i \in [k]\}$, $\Psi_G = \{\psi_{G,i} : i \in \{0,\ldots,k\}\}$ and $\Psi_N = \{\psi_{N,i,j} : i,j \in \{0,\ldots,k\}, i \neq j\}$, where $\psi_{P,i}(a) = \mathbf{1}\left[a < i\right]$, $\psi_{G,i}(a) = \mathbf{1}\left[a = i\right]$ and $\psi_{N,i,j}(a)$ equals $1, 0, *$ if $a = i, a = j, a \notin \{i,j\}$ respectively, define the* Pollard pseudo-dimension $\Psi_P\text{-dim}\left(V\right)$*, the* Graph dimension $\Psi_G\text{-dim}\left(V\right)$ *and the* Natarajan dimension $\Psi_N\text{-dim}\left(V\right)$*.*

## 2.3 The one-inclusion prediction strategy

A subset of the $n$-cube—the projection of some $\mathcal{F}$—induces the one-inclusion graph, which underlies a natural prediction strategy. The following definition generalizes this to a subset of $\{0,\ldots,k\}^n$.

**Definition 2.7** *The* one-inclusion hypergraph $\mathcal{G}\left(V\right) = (V,E)$ of $V \subseteq \{0,\ldots,k\}^n$ *is the undirected graph with vertex-set $V$ and hyperedge-set $E$ of maximal (with respect to inclusion) sets of pairwise hamming-$1$ separated vertices.*

---

**Algorithm 1** The deterministic multiclass one-inclusion prediction strategy $Q_{\mathcal{G},\mathcal{F}}$

---

**Given:** $\mathcal{F} \subseteq \{0,\ldots,k\}^{\mathcal{X}}$, $\mathrm{sam}\left((x_1,\ldots,x_{n-1}),f\right) \in (\mathcal{X} \times \{0,1\})^{n-1}$, $x_n \in \mathcal{X}$
**Returns:** a prediction of $f(x_n)$

$V \quad \longleftarrow \Pi_{\mathbf{x}}\left(\mathcal{F}\right)$ ;
$G \quad \longleftarrow \mathcal{G}\left(V\right)$ ;
$\overrightarrow{G} \quad \longleftarrow$ orient $G$ to minimize the maximum outdegree ;
$V_{\text{space}} \longleftarrow \{\mathbf{v} \in V \mid v_1 = f(x_1),\ldots,v_{n-1} = f(x_{n-1})\}$ ;
**if** $V_{\text{space}} = \{\mathbf{v}\}$ **then return** $v_n$ ;
**else return** the $n^{\text{th}}$ component of the head of hyperedge $V_{\text{space}}$ in $\overrightarrow{G}$ ;

---

The one-inclusion graph's prediction strategy $Q_{\mathcal{G},\mathcal{F}}$ [4] immediately generalizes to the multiclass prediction strategy described by Algorithm 1. For the remainder of this and Section 4 we will restrict our discussion to the $k = 1$ case, on which the following main result of [4] focuses.

**Theorem 2.8 (Theorem 2.3 [4])** $\hat{M}_{Q_{\mathcal{G},\mathcal{F}},\mathcal{F}}(n) \leq \frac{\mathrm{VC}(\mathcal{F})}{n}$ *for every concept class $\mathcal{F}$ and $n > 1$.*

A lower bound in [6] showed that the one-inclusion strategy's performance is optimal within a factor of $1 + o(1)$. Replacing orientation with a distribution over each edge induces a *randomized strategy* $Q_{\mathcal{G}rand,\mathcal{F}}$. The key to proving Theorem 2.8 is the following.

**Lemma 2.9 (Lemma 2.4 [4])** *For any $n \in \mathbb{N}$ and $V \subseteq \{0,1\}^n$, $\mathrm{dens}\left(\mathcal{G}\left(V\right)\right) \leq \mathrm{VC}(V)$.*

An elegant proof of this deep result, due to Haussler [3], uses *shifting*. Consider any $s \in [n]$, $\mathbf{v} \in V$ and let $S_s(\mathbf{v}; V)$ be $\mathbf{v}$ shifted along $s$: if $v_s = 0$, or if $v_s = 1$ and there exists some $\mathbf{u} \in V$ differing to $\mathbf{v}$ only in the $s^{\text{th}}$ coordinate, then $S_s(\mathbf{v}; V) = \mathbf{v}$; otherwise $\mathbf{v}$ shifts down—its $s^{\text{th}}$ coordinate is decreased from 1 to 0. The entire family $V$ can be shifted to $S_s(V) = \{S_s(\mathbf{v}; V) \mid \mathbf{v} \in V\}$ and this shifted vertex-set induces $S_s(E)$ the edge-set of $\mathcal{G}(S_s(V))$, where $(V, E) = \mathcal{G}(V)$.

**Definition 2.10** *Let $I \subseteq [n]$. We call a subset $V \subseteq \{0,1\}^n$ $I$-closed-below if $S_s(V) = V$ for all $s \in I$. If $V$ is $[n]$-closed-below then we call it* closed-below.

A number of properties of shifting follow relatively easily:

$$
\begin{array}{rcll}
|S_s(V)| & = & |V| \ , & \text{by the injectivity of } S_s(\,\cdot\,; V) \qquad\qquad (1) \\
\mathrm{VC}(S_s(V)) & \leq & \mathrm{VC}(V) \ , & \text{as } S_s(V) \text{ shatters } I \subseteq [n] \Rightarrow V \text{ shatters } I \qquad (2) \\
|E| & \leq & |V| \cdot \mathrm{VC}(V) \ , & \text{as } V \text{ closed-below} \Rightarrow \max_{\mathbf{v} \in V} \|\mathbf{v}\|_{l_1} \leq \mathrm{VC}(V) \ (3) \\
|S_s(E)| & \geq & |E| \ , & \text{by cases} \qquad\qquad (4)
\end{array}
$$

$$\exists T \in \mathbb{N}, \mathbf{s} \in [n]^T \quad \text{s.t.} \quad S_{s_T}(\ldots S_{s_1}(V)) \text{ is closed-below (a fixed-point)} \ . \qquad (5)$$

Properties (1–2) and the justification of (3) together imply Sauer's lemma; Properties (1–5) lead to

$$\frac{|E|}{|V|} \ \leq \ \ldots \ \leq \ \frac{|S_{s_T}(\ldots S_{s_1}(E))|}{|S_{s_T}(\ldots S_{s_1}(V))|} \ \leq \ \mathrm{VC}(S_{s_T}(\ldots S_{s_1}(V))) \ \leq \ \ldots \ \leq \ \mathrm{VC}(V) \quad .$$

# 3 Shatter-invariant shifting

While [3] shifts to bound density, the number of edges can increase *and* the VC-dimension can decrease—both contributing to the observed gap between graph density and capacity. The next result demonstrates that shifting can in fact be controlled to preserve VC-dimension.

**Lemma 3.1** *Consider arbitrary $n \in \mathbb{N}$, $I \subseteq [n]$ and $V \subseteq \{0,1\}^n$ that shatters $I$. There exists a finite sequence $s_1, \ldots, s_T$ in $[n]$ such that each $V_t = S_{s_t}(\ldots S_{s_1}(V))$ shatters $I$ and $V_T$ is closed-below. In particular $\mathrm{VC}(V_T) = \mathrm{VC}(V_{T-1}) = \ldots = \mathrm{VC}(V)$.*

**Proof:** $\Pi_I(\cdot)$ is invariant to shifting on $\overline{I} = [n] \backslash I$. So some finite number of shifts on $\overline{I}$ will produce a $\overline{I}$-closed-below family $W$ that shatters $I$. Hence $W$ must contain representatives for each element of $\{0,1\}^{|I|}$ (embedded at $I$) with components equal to 0 outside $I$. Thus the shattering of $I$ is invariant to the shifting of $W$ on $I$, so that a finite number of shifts on $I$ produces an $I$-closed-below $W'$ that shatters $I$. Repeating the process a finite number of times until no non-trivial shifts are made produces a closed-below family that shatters $I$. The second claim follows from (2). ∎

# 4 Tightly bounding graph density by symmetrization

Kuzmin and Warmuth [5] introduced $D_n^d$ as a potential bound on the graph density of maximum classes. We begin with properties of $D_n^d$, a technical lemma and then proceed to the main result.

**Definition 4.1** *Define $D_n^d = \frac{n \binom{n-1}{\leq d-1}}{\binom{n}{\leq d}}$ for all $n \in \mathbb{N}$ and $d \in [n]$. Denote by $V_n^d$ the VC-dimension $d$ closed-below subset of $\{0,1\}^n$ equal to the union of all $\binom{n}{d}$ closed-below embedded $d$-cubes.*

**Lemma 4.2** $D_n^d$
- (i) *equals the graph density of $V_n^d$ for each $n \in \mathbb{N}$ and $d \in [n]$;*
- (ii) *is strictly upper-bounded by $d$, for all $n$;*
- (iii) *equals $\frac{d}{2}$ for all $n = d \in \mathbb{N}$;*
- (iv) *is strictly monotonic increasing in $d$ (with $n$ fixed);*
- (v) *is strictly monotonic increasing in $n$ (with $d$ fixed); and*
- (vi) *limits to $d$ as $n \to \infty$.*

**Proof:** By counting, for each $d \le n < \infty$, the density of $\mathcal{G}\left(V_n^d\right)$ equals $D_n^d$:

$$\frac{\left|\mathrm{E}\left(\mathcal{G}\left(V_n^d\right)\right)\right|}{\left|V_n^d\right|} = \frac{\sum_{i=1}^{d} i\binom{n}{i}}{\sum_{i=0}^{d}\binom{n}{i}} = \frac{n\sum_{i=0}^{d-1}\frac{i+1}{n}\binom{n}{i+1}}{\binom{n}{\le d}} = \frac{n\sum_{i=0}^{d-1}\binom{n-1}{i}}{\binom{n}{\le d}} = \frac{n\binom{n-1}{\le d-1}}{\binom{n}{\le d}}$$

proving (i). Since for all $A, B, C, D > 0$, $\frac{A}{B} < \frac{A+C}{B+D}$ iff $\frac{A}{B} < \frac{C}{D}$, it is sufficient for (iv) to prove that $D_n^{d-1} < \frac{n\binom{n-1}{d-1}}{\binom{n}{d}}$. By (i) and Lemma 2.9 $D_n^d \le d$, and so

$$D_n^{d-1} \le d - 1 < d = \frac{n \cdot (n-1)!}{n!}\frac{(n-d)!}{(n-d)!}\frac{d!}{(d-1)!} = \frac{n\frac{(n-1)!}{(n-d)!(d-1)!}}{\frac{n!}{(n-d)!d!}} = \frac{n\binom{n-1}{d-1}}{\binom{n}{d}} \quad .$$

Monotonicity in $d$, (i) and Lemma 2.9 together prove (ii). Properties (iii,v–vi) are proven in [8]. ∎

**Lemma 4.3** *For arbitrary* $U, V \subseteq \{0,1\}^n$ *with* $\mathrm{dens}\left(\mathcal{G}\left(V\right)\right) \ge \rho > 0$, $|U| \le |V|$ *and* $\left|\mathrm{E}\left(\mathcal{G}\left(U\right)\right)\right| \ge \left|\mathrm{E}\left(\mathcal{G}\left(V\right)\right)\right|$, *if* $\mathrm{dens}\left(\mathcal{G}\left(U \cap V\right)\right) < \rho$ *then* $\mathrm{dens}\left(\mathcal{G}\left(U \cup V\right)\right) > \rho$.

**Proof:** If $\mathcal{G}\left(U \cap V\right)$ has density less than $\rho$ then

$$\frac{\left|\mathrm{E}\left(\mathcal{G}\left(U \cup V\right)\right)\right|}{\left|U \cup V\right|} \ge \frac{\left|\mathrm{E}\left(\mathcal{G}\left(U\right)\right)\right| + \left|\mathrm{E}\left(\mathcal{G}\left(V\right)\right)\right| - \left|\mathrm{E}\left(\mathcal{G}\left(U \cap V\right)\right)\right|}{|U| + |V| - |U \cap V|}$$

$$\ge \frac{2\left|\mathrm{E}\left(\mathcal{G}\left(V\right)\right)\right| - \left|\mathrm{E}\left(\mathcal{G}\left(U \cap V\right)\right)\right|}{2|V| - |U \cap V|}$$

$$> \frac{2\rho|V| - \rho|U \cap V|}{2|V| - |U \cap V|} = \rho$$

∎

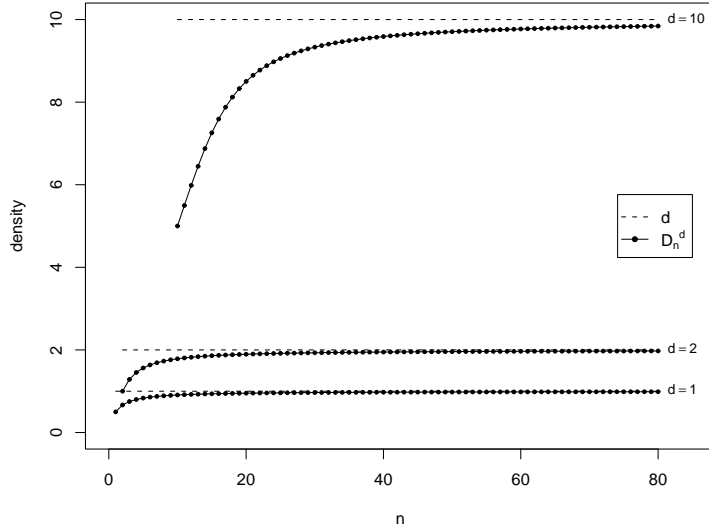

Figure 1: The improved graph density bound of Theorem 4.4. The density bounding $D_n^d$ is plotted (dotted solid) alongside the previous best $d$ (dashed), for each $d \in \{1, 2, 10\}$.

**Theorem 4.4** *Every family* $V \subseteq \{0,1\}^n$ *with* $d = \mathrm{VC}(V)$ *has* $(V, E) = \mathcal{G}\left(V\right)$ *with graph density*

$$\frac{|E|}{|V|} \le D_n^d < d \quad . \tag{6}$$

*For* $n \in \mathbb{N}$ *and* $d \in [n]$, $V_n^d$ *is the unique closed-below VC-dimension* $d$ *subset of* $\{0,1\}^n$ *meeting (6) with equality. A VC-dimension* $d$ *family* $V \subseteq \{0,1\}^n$ *meets (6) with equality only if* $V$ *is maximum.*

**Proof:** Allow a permutation $g \in S_n$ to act on vector $\mathbf{v} \in \{0,1\}^n$ and family $V \subseteq \{0,1\}^n$ by $g(\mathbf{v}) = \left(v_{g(1)}, \dots, v_{g(n)}\right)$ and $g(V) = \{g(\mathbf{v}) \mid \mathbf{v} \in V\}$; and define $S_n(V) = \bigcup_{g \in S_n} g(V)$. Note

that a closed-below VC-dimension $d$ family $V \subseteq \{0,1\}^n$ satisfies $S_n(V) = V$ iff $V = V_n^d$, as $\mathrm{VC}(V) \geq d$ implies $V$ contains an embedded $d$-cube, invariance to $S_n$ implies further that $V$ contains all $\binom{n}{d}$ such cubes, and $\mathrm{VC}(V) \leq d$ implies that $V \subseteq V_n^d$. Consider now any

$$V_{n,d}^* \in \arg\min \left\{ |U| \;\middle|\; U \in \underset{\{U \subseteq \{0,1\}^n | \mathrm{VC}(U) \leq d, U \text{ closed-below}\}}{\arg\max} \mathrm{dens}\left(\mathcal{G}\left(U\right)\right) \right\} .$$

For the purposes of contradiction assume that $V_{n,d}^* \neq g(V_{n,d}^*)$ for some $g \in S_n$. Then if $\mathrm{dens}\left(\mathcal{G}\left(V_{n,d}^* \cap g(V_{n,d}^*)\right)\right) \geq \mathrm{dens}\left(\mathcal{G}\left(V_{n,d}^*\right)\right)$ then $V_{n,d}^*$ would not have been selected above (i.e. a closed-below family at least as small and dense as $V_{n,d}^* \cap g(V_{n,d}^*)$ would have been chosen). Thus $\mathrm{dens}\left(\mathcal{G}\left(V_{n,d}^* \cup g(V_{n,d}^*)\right)\right) > \mathrm{dens}\left(\mathcal{G}\left(V_{n,d}^*\right)\right)$ by Lemma 4.3. But then again $V_{n,d}^*$ would not have been selected (i.e. a distinct family at least as dense as $V_{n,d}^* \cup g(V_{n,d}^*)$ would have been selected instead, since every vector in this union contains no more than $d$ 1's). Hence $V_{n,d}^* = S_n(V_{n,d}^*)$ and so $V_{n,d}^* = V_n^{d'}$ and by Lemma 4.2.(i) $\mathrm{dens}\left(\mathcal{G}\left(V_{n,d}^*\right)\right) = D_n^{d'}$, for $d' = \mathrm{VC}(V_{n,d}^*) \leq d$. But by Lemma 4.2.(iv) this implies that $d = d'$ and (6) is true for all closed-below families; $V_n^d$ uniquely maximizes density amongst all closed-below VC-dimension $d$ families in the $n$-cube.

For an arbitrary $V \subseteq \{0,1\}^n$ with $d = \mathrm{VC}(V)$ consider any of its closed-below fixed-point (cf. (5)), $W \subseteq \{0,1\}^n$. Noting that $\mathrm{VC}(W) \leq d$ and $\mathrm{dens}\left(\mathcal{G}\left(V\right)\right) \leq \mathrm{dens}\left(\mathcal{G}\left(W\right)\right)$ by (2) and (1) & (4) respectively, the bound (6) follows directly for $V$. Furthermore if we shift to preserve VC-dimension then $\mathrm{VC}(W) = d$ while still $|V| = |W|$. And since $\mathrm{dens}\left(\mathcal{G}\left(W\right)\right) = D_n^d$ only if $W = V_n^d$, it follows that $V$ maximizes density amongst all VC-dimension $d$ families in the $n$-cube, with $\mathrm{dens}\left(\mathcal{G}\left(V\right)\right) = D_n^d$, only if it is maximum. ∎

Theorem 4.4 improves on the VC-dimension density bound of Lemma 2.9 for low sample sizes (see Figure 1). This new result immediately implies the following one-inclusion mistake bounds.

**Theorem 4.5** *Consider any $n \in \mathbb{N}$ and $\mathcal{F} \subseteq \{0,1\}^{\mathcal{X}}$ with $\mathrm{VC}(\mathcal{F}) = d < \infty$. Then $\hat{M}_{Q_{\mathcal{G},\mathcal{F}},\mathcal{F}}(n) \leq \left\lceil D_n^d \right\rceil / n$ and $\hat{M}_{Q_{\mathcal{G}rand,\mathcal{F}},\mathcal{F}}(n) \leq D_n^d / n$.*

For small $d$, $n^*(d) = \min\left\{ n \geq d \mid d = \left\lceil D_n^d \right\rceil \right\}$—the first $n$ for which the new and old deterministic one-inclusion mistake bounds coincide—appears to remain very close to $2.96d$. The randomized strategy's mistake bound of Theorem 4.5 offers a strict improvement over that of [4].

# 5  Bounds for multiclass prediction

As in the $k = 1$ case, the key to developing the multiclass one-inclusion mistake bound is in bounding hypergraph density. We proceed by shifting a graph induced by the one-inclusion hypergraph.

**Theorem 5.1** *For any $k, n \in \mathbb{N}$ and $V \subseteq \{0, \ldots, k\}^n$, the one-inclusion hypergraph $(V, E) = \mathcal{G}\left(V\right)$ satisfies $\frac{|E|}{|V|} \leq \Psi_{\mathrm{P}}\text{-dim}\left(V\right)$.*

**Proof:** We begin by replacing the hyperedge structure $E$ with a related edge structure $E'$. Two vertices $\mathbf{u}, \mathbf{v} \in V$ are connected in the graph $(V, E')$ iff there exists an $i \in [n]$ such that $\mathbf{u}, \mathbf{v}$ differ only at $i$ and no $\mathbf{w} \in V$ exists such that $u_i < w_i < v_i$ and $w_j = u_j = v_j$ on $[n] \backslash \{i\}$. Trivially

$$\frac{|E|}{|V|} \leq \frac{|E'|}{|V|} \leq \frac{k|E|}{|V|} . \tag{7}$$

Consider now shifting vertex $\mathbf{v} \in V$ at shift label $t \in [k]$ along shift coordinate $s \in [n]$ by

$$\begin{aligned}
S_{s,t}(\mathbf{v}; V) &= \mathbf{v}^{s(v_s')} \\
\text{where} \\
\mathbf{v}^{s(i)} &= (v_1, \ldots, v_{s-1}, i, v_{s+1}, \ldots, v_n) \quad \text{for } i \in \{0, \ldots, k\} \\
v_s' &= \begin{cases} \min\left\{ x \in \{0, \ldots, v_s\} \mid \mathbf{v}^{s(x)} \notin V \text{ or } x = v_s \right\} & \text{if } v_s = t \\ v_s & \text{o.w.} \end{cases}
\end{aligned}$$

We shift $V$ on $s$ at $t$ as usual; we shift $V$ on $s$ alone by bubbling vertices down to fill gaps below:

$$\begin{aligned} S_{s,t}(V) &= \{S_{s,t}(\mathbf{v};V) \mid \mathbf{v} \in V\} \\ S_s(V) &= S_{s,k}(S_{s,k-1}(\ldots S_{s,1}(V))) \ . \end{aligned}$$

Let $S_s(E')$ denote the *edge*-set induced by $S_s(V)$. $S_s$ on a vertex-set is injective implying that

$$|S_s(V)| = |V| \ . \tag{8}$$

Consider any $\{\mathbf{u},\mathbf{v}\} \in E'$ with $i \in [n]$ denoting the index on which $\mathbf{u}, \mathbf{v}$ differ. If $i = s$ then no other vertex $\mathbf{w} \in V$ can come between $\mathbf{u}$ and $\mathbf{v}$ during shifting by construction of $E'$, so $\{S_s(\mathbf{u};V), S_s(\mathbf{v};V)\} \in S_s(E')$. Now suppose that $i \neq s$. If both vertices shift down by the same number of labels then they remain connected in $S_s(E')$. Otherwise assume WLOG that $S_s(\mathbf{u};V)_s < S_s(\mathbf{v};V)_s$ then the shifted vertices will lose their edge, however since $v_s$ did not shift down to $S_s(\mathbf{u};V)_s$ there must have been some $\mathbf{w} \in V$ different to $\mathbf{u}$ on $\{i,s\}$ such that $w_s < v_s$ with $S_s(\mathbf{w};V)_s = S_s(\mathbf{u};V)_s$. Thus $S_s(\mathbf{w};V), S_s(\mathbf{u};V)$ differ only on $\{i\}$ and a new edge $\{S_s(\mathbf{w};V), S_s(\mathbf{u};V)\}$ is in $S_s(E')$ that was not in $E'$ (otherwise $\mathbf{u}$ would not have shifted). Thus

$$|S_s(E')| \geq |E'| \ . \tag{9}$$

Suppose that $I \subseteq [n]$ is $\Psi_P$-shattered by $S_s(V)$. If $s \notin I$ then $\Pi_I(S_s(V)) = \Pi_I(V)$ and $I$ is $\Psi_P$-shattered by $V$. If $s \in I$ then $V$ $\Psi_P$-shatters $I$. Witnesses of $S_s(V)$'s $\Psi_P$-shattering of $I$ equal to 1 at $s$, taking each value in $\{0,1\}^{|I|-1}$ on $I\backslash\{s\}$, were not shifted and so are witnesses for $V$; since these vertices were not shifted they were blocked by vertices of $V$ of equal values on $I\backslash\{s\}$ but equal to 0 at $s$, these are the remaining half of the witnesses of $V$'s $\Psi_P$-shattering of $I$. Thus

$$S_s(V) \ \Psi_P\text{-shatters } I \subseteq [n] \quad \Rightarrow \quad V \ \Psi_P\text{-shatters } I \ . \tag{10}$$

In a finite number of shifts starting from $(V, E')$, a closed-below family $W$ with induced edge-set $F$ will be reached. If $I \subseteq [n]$ is $\Psi_P$-shattered by $W$ and $|I| = d = \Psi_P\text{-dim}(W)$, then since $W$ is closed-below the translation vector $(\psi_{P,1}, \ldots, \psi_{P,1})(\cdot) = (\mathbf{1}[\cdot < 1], \ldots, \mathbf{1}[\cdot < 1])$ must witness this shattering. Hence each $\mathbf{w} \in W$ has at most $d$ non-zero components. Counting edges in $F$ by upper-adjoining vertices we have proved that

$$(V, E') \text{ finitely shifts to closed-below graph } (W, F) \quad \text{s.t.} \quad |F| \leq |W| \cdot \Psi_P\text{-dim}(W) \ . \tag{11}$$

Combining properties (7)–(11) we have that $\frac{|E|}{|V|} \leq \frac{|E'|}{|V|} \leq \frac{|F|}{|W|} \leq \Psi_P\text{-dim}(W) \leq \Psi_P\text{-dim}(V)$. ∎

The remaining arguments from the $k = 1$ case of [4, 3] now imply the multiclass mistake bound.

**Theorem 5.2** *Consider any $k, n \in \mathbb{N}$ and $\mathcal{F} \subseteq \{0, \ldots, k\}^{\mathcal{X}}$ with $\Psi_P\text{-dim}(\mathcal{F}) < \infty$. The multiclass one-inclusion prediction strategy satisfies $\hat{M}_{Q_{\mathcal{G},\mathcal{F}},\mathcal{F}}(n) \leq \Psi_P\text{-dim}(\mathcal{F})/n$.*

## 5.1 A lower bound

We now show that the preceding multiclass mistake bound is optimal to within a $O(\log k)$ factor, noting that $\Psi_N$ is smaller than $\Psi_P$ by at most such a factor [1, Theorem 10].

**Definition 5.3** *We call a family $\mathcal{F} \subseteq \{0, \ldots, k\}^{\mathcal{X}}$ trivial if either $|\mathcal{F}| = 1$ or there exist no $x_1, x_2 \in \mathcal{X}$ and $f_1, f_2 \in \mathcal{F}$ such that $f_1(x_1) \neq f_2(x_1)$ and $f_1(x_2) = f_2(x_2)$.*

**Theorem 5.4** *Consider any deterministic or randomized prediction strategy $Q$ and any $\mathcal{F} \subseteq \{0, \ldots, k\}^{\mathcal{X}}$ that has $2 \leq \Psi_N\text{-dim}(\mathcal{F}) < \infty$ or is non-trivial with $\Psi_N\text{-dim}(\mathcal{F}) < 2$. Then for all $n > \Psi_N\text{-dim}(\mathcal{F})$, $\hat{M}_{Q,\mathcal{F}}(n) \geq \max\{1, \Psi_N\text{-dim}(\mathcal{F}) - 1\}/(2en)$.*

**Proof:** Following [2], we use the probabilistic method to prove the existence of a target in $\mathcal{F}$ for which prediction under a distribution $P$ supported by a $\Psi_N$-shattered subset is hard. Consider $d = \Psi_N\text{-dim}(\mathcal{F}) \geq 2$ with $n > d$. Fix a $\mathcal{Z} = \{z_1, \ldots, z_d\}$ $\Psi_N$-shattered by $\mathcal{F}$ and then a subset $\mathcal{F}_{\mathcal{Z}} \subseteq \mathcal{F}$ of $2^d$ functions that $\Psi_N$-shatters $\mathcal{Z}$. Define a distribution $P$ on $\mathcal{X}$ by $P(\{z_i\}) = n^{-1}$ for each $i \in [d-1]$, $P(\{z_d\}) = 1 - (d-1)n^{-1}$ and $P(\{x\}) = 0$ for all $x \in \mathcal{X}\backslash\mathcal{Z}$. Observe that $\Pr_{P^n}(\forall i \in [n-1], X_n \neq X_i) \geq \Pr_{P^n}(X_n \neq z_d, \forall i \in [n-1], X_n \neq X_i) =$

$\frac{d-1}{n}\left(1-\frac{1}{n}\right)^{n-1} \geq \frac{d-1}{en}$. For any $f \in \mathcal{F}_{\mathcal{Z}}$ and $\mathbf{x} \in \mathcal{Z}^n$ with $x_n \neq x_i$ for all $i \in [n-1]$, exactly half of the functions in $\mathcal{F}_{\mathcal{Z}}$ consistent with $\mathrm{sam}\left((x_1,\dots,x_{n-1}),f\right)$ output some $i \in \{0,\dots,k\}$ on $x_n$ and the remaining half output some $j \in \{0,\dots,k\}\backslash\{i\}$. Thus $\mathbb{E}_{\mathrm{Unif}(\mathcal{F}_{\mathcal{Z}})}\left[\mathbf{1}\left[Q(\mathrm{sam}\left((x_1,\dots,x_{n-1},F)\right),x_n) \neq F(x_n)\right]\right] = 0.5$ for such an $\mathbf{x}$ and so

$$\hat{M}_{Q,\mathcal{F}} \geq \hat{M}_{Q,\mathcal{F}_{\mathcal{Z}}} \geq \mathbb{E}_{\mathrm{Unif}(\mathcal{F}_{\mathcal{Z}})\times P^n}\left[\mathbf{1}\left[Q(\mathrm{sam}\left((X_1,\dots,X_{n-1},F)\right),X_n) \neq F(X_n)\right]\right] \geq \frac{d-1}{2en}.$$

The similar case of $d < 2$ is omitted here and shows that there is a distribution $P$ on $\mathcal{X}$ and function $f \in \mathcal{F}$ such that $\mathbb{E}_{P^n}\left[\mathbf{1}\left[Q(\mathrm{sam}\left((X_1,\dots,X_{n-1}),f\right),X_n) \neq f(X_n)\right]\right] \geq (2en)^{-1}$. ∎

# 6 Conclusions and open problems

In this paper we have developed new shifting machinery and tightened the binary one-inclusion mistake bound from $d/n$ to $D_n^d/n$ ($\lceil D_n^d \rceil/n$ for the deterministic strategy) representing a solid improvement for $d \approx n$. We have described the multiclass generalization of the prediction learning model and derived a mistake bound for the multiclass one-inclusion prediction strategy that improves on previous PAC-based expected risk bounds by $O(\log n)$ and that is within $O(\log k)$ of optimal.

Here shifting with invariance to the shattering of a single set was described, however we are aware of invariance to more complex shatterings. Another serious application of shatter-invariant shifting, to appear in a sequel to this paper, is to the study of the cubical structure of maximum and maximal classes with connections to the compressibility conjecture of [7]. While Theorem 4.4 resolves one conjecture of Kuzmin & Warmuth [5], the remainder of the conjectured correctness proof for the Peeling compression scheme is known to be false [8].

The symmetrization method of Theorem 4.4 can be extended over subgroups $G \subset S_n$ to gain tighter density bounds. Just as the $S_n$-invariant $V_n^d$ is the maximizer of density among all closed-below $V \subseteq V_n^d$, there exist $G$-invariant families that maximize the density over all of *their* sub-families.

In addition to Theorem 5.2 we have also proven the following special case in terms of $\Psi_G$; it is open as to whether this generalizes to $n \in \mathbb{N}$. While a general $\Psi_G$-based bound would allow direct comparison with the PAC-based expected risk bound, it should also be noted that $\Psi_P$ and $\Psi_G$ are in fact incomparable—neither $\Psi_G \leq \Psi_P$ nor $\Psi_P \leq \Psi_G$ singly holds for all classes [1, Theorem 1].

**Lemma 6.1 ([8])** *For any $k \in \mathbb{N}$ and family $V \subseteq \{0,\dots,k\}^2$, $\mathrm{dens}\left(\mathcal{G}\left(V\right)\right) \leq \Psi_{\mathrm{G}}\text{-}\dim\left(V\right)$.*

## Acknowledgments

We gratefully acknowledge the support of the NSF under award DMS-0434383.

## References

[1] Ben-David, S., Cesa-Bianchi, N., Haussler, D., Long, P. M.: Characterizations of learnability for classes of $\{0,\dots,n\}$-valued functions. *Journal of Computer and System Sciences*, **50**(1) (1995) 74–86

[2] Ehrenfeucht, A., Haussler, D., Kearns, M., Valiant, L.: A general lower bound on the number of examples needed for learning. *Information and Computation*, **82**(3) (1989) 247–261

[3] Haussler, D.: Sphere packing numbers for subsets of the boolean $n$-cube with bounded Vapnik-Chervonenkis dimension. *Journal of Combinatorial Theory (A)* **69**(2) (1995) 217–232

[4] Haussler, D., Littlestone, N., Warmuth, M. K.: Predicting $\{0,1\}$ functions on randomly drawn points. *Information and Computation*, **115**(2) (1994) 284–293

[5] Kuzmin, D., Warmuth, M. K.: Unlabeled compression schemes for maximum classes. *Journal of Machine Learning Research* (2006) to appear

[6] Li, Y., Long, P. M., Srinivasan, A.: The one-inclusion graph algorithm is near optimal for the prediction model of learning. *IEEE Transactions on Information Theory*, **47**(3) (2002) 1257–1261

[7] Littlestone, N., Warmuth, M. K.: Relating data compression and learnability. Unpublished manuscript, http://www.cse.ucsc.edu/~manfred/pubs/lrnk-olivier.pdf (1986)

[8] Rubinstein, B. I. P., Bartlett, P. L., Rubinstein, J. H.: Shifting: One-Inclusion Mistake Bounds and Sample Compression. Technical report, EECS Department, UC Berkeley (2007) to appear

[9] Sauer, N.: On the density of families of sets. *Journal of Combinatorial Theory (A)*, **13** (1972) 145–147
